# Navigating through Temporal Difference

**Peter Dayan**
Centre for Cognitive Science & Department of Physics
University of Edinburgh
2 Buccleuch Place, Edinburgh EH8 9LW
dayan@cns.ed.ac.uk

## Abstract

Barto, Sutton and Watkins [2] introduced a grid task as a didactic example of temporal difference planning and asynchronous dynamical programming. This paper considers the effects of changing the coding of the input stimulus, and demonstrates that the self-supervised learning of a particular form of hidden unit representation improves performance.

## 1  INTRODUCTION

Temporal difference (TD) planning [6, 7] uses prediction for control. Consider an agent moving around a finite grid such as the one in figure 1 (the agent is incapable of crossing the barrier) trying to reach a goal whose position it does not know. If it can predict how far away from the goal it is at the current step, and how far away from the goal it is at the next step, after making a move, then it can decide whether or not that move was helpful or harmful. If, in addition, it can record this fact, then it can learn how to navigate to the goal. This generation of actions from predictions is closely related to the mechanism of dynamical programming.

TD is used to learn the predictions in the first place. Consider the agent moving around randomly on the grid, receiving a negative reinforcement of $-1$ for every move it makes apart from moves which take it onto the goal. In this case, if it can estimate from every location it visits, how much reinforcement (discounted by how soon it arrives) it will get before it next reaches the goal, it will be predicting how far away it is, based on the random method of selecting actions. TD's mechanism of learning is to force the predictions to be consistent; the prediction from location a should be $-1$ more than the average of the predictions from the locations that can be reached in one step (hence the extra $-1$ reinforcement) from a.

If the agent initially selects each action with the same probability, then the estimate of future reinforcement from a will be monotonically related to how many steps a is away from the goal. This makes the predictions useful for criticising actions as above. In practice, the agent will modify its actions according to this criticism at the same time as learning the predictions based on those actions.

Barto, Sutton and Watkins [2] develop this example, and show how the TD mechanism coupled with a punctate representation of the stimulus (referred to as $\mathcal{R}_{BSW}$ below) finds the optimal paths to the goal. $\mathcal{R}_{BSW}$ ignores the cues shown in figure 1, and devotes one input unit to each location on the grid, which fires if and only if the agent is at that place.

TD methods can however work with more general codes. Section 2 considers alternative representations, including ones that are sensitive to the orientation of the agent as it moves through the grid, and section 3 looks at a restricted form of latent learning – what the agent can divine about its environment in the absence of reinforcement. Both techniques can improve the speed of learning.

## 2   ALTERNATE REPRESENTATIONS

Stimulus representations, the means by which the agent finds out from the environment where it is, can be classified along two dimensions; whether they are punctate or distributed, and whether they are directionally sensitive or in register with the world.

Over most of the grid, a 'sensible' distributed representation, such as a coarse-coded one, would be expected to make learning faster, as information about the value and action functions could be shared across adjacent grid points. There are points of discontinuity in the actions, as in the region above the right hand arm of the barrier, but they are few. In his PhD thesis [9], Watkins considers a rather similar problem to that in figure 1, and solves it using his variant of TD, Q-learning, based on a CMAC [1] coarse-coded representation of the space. Since his agent moves in a continuous bounded space, rather than being confined merely to discrete grid points, something of this sort is anyway essential. After the initial learning, Watkins arbitrarily makes the agent move ten times more slowly in a closed section of the space. This has a similar effect to the barrier in inducing a discontinuity in the action space. Despite the CMACs forcing the system to share information across such discontinuities, they were able to learn the task quickly.

The other dimension over which representations may vary involves the extent to which they are sensitive to the direction in which the agent is facing. This is of interest if the agent must construe its location from the cues around the grid. In this case, rather than moving North, South, East or West, which are actions registered with the world, the agent should only move Ahead, Left or Right (Behind is disabled as an additional constraint), whose effects are also orientation dependent. This, together with the fact that the representation will be less compact (ie having a larger input dimensionality) should make learning slower. Dynamical programming and its equivalents are notoriously subject to Bellman's curse of dimensionality, an engineering equivalent of exponential explosion in search.

Table 1 shows four possible representations classified along these two dimensions.

| Directionally | Coarseness | |
| --- | --- | --- |
| | Punctate | Distributed |
| Sensitive | $\mathcal{R}_{4X}$ | $\mathcal{R}_A$ |
| Insensitive | $\mathcal{R}_{BSW}$ | $\mathcal{R}_{CMAC}$ |

Table 1: Representations.

$\mathcal{R}_{BSW}$ is the representation Barto, Sutton and Watkins used. $\mathcal{R}_{4X}$ is punctate and directionally sensitive - it devotes four units to every grid point, one of which fires for each possible orientation of the agent. $\mathcal{R}_{CMAC}$, the equivalent of Watkins' representation, was not simulated, because its capabilities would not differ markedly from those of the mapping-based representation developed in the next section.

$\mathcal{R}_A$ is rather different from the other representations; it provides a test of a representation which is more directly associated with the sensory information that might be available directly from the cues. Figure 2 shows how $\mathcal{R}_A$ works. Various identifiable cues, $C_1 \ldots C_c$ ($c = 7$ in the figure) are scattered around the outside of the grid, and the agent has a fictitious 'retina' which rotates with it. This retina is divided into a number of angular buckets (8 in the figure), and each bucket has $c$ units, the $i^{th}$ one of which responds if the cue $C_i$ is visible in that bucket. This representation is clearly directionally sensitive (if the agent is facing a different way, then so is its retina, and so no cue will be visible in the same bucket as it was before), and also distributed, since in general more than one cue will be visible from every location.

Note that there is no restriction on the number of units that can fire in each bucket at any time - more than one will fire if more than one cue is visible there. Also, under the present system $\mathcal{R}_A$ will in general not work if its coding is ambiguous - grid points must be distinguishable. Finally, it should be clear that $\mathcal{R}_A$ is not biologically plausible.

Figure 3 shows the learning curves for the three representations simulated. Each point is generated by switching off the learning temporarily after a certain number of iterations, starting the agent from everywhere in the grid, and averaging how many steps it takes in getting to the goal over and above the minimum necesary. It is apparent that $\mathcal{R}_{4X}$ is substantially worse, but, surprisingly, that $\mathcal{R}_A$ is actually better than $\mathcal{R}_{BSW}$. This implies that the added advantage of its distributed nature more than outweighs its disadvantages of having more components and being directionally sensitive.

One of the motivations behind studying alternate representations is the experimental findings on *place* cells in the hippocampi of rats (amongst other species). These are cells that fire only when the rat is at a certain location in its environment. Although their existence has led to many hypotheses about rat cognitive mapping (see [5] for a substantial discussion of place cells and mapping), it is important to note that even with a map, there remains the computationally intensive problem of navigation addressed, in this paper, by TD. $\mathcal{R}_A$, being closely related to the input stimuli is quite unlike a place cell code - the other representations all bear some similarities.

# 3   GOAL-FREE LEARNING

One of the problems with the TD system as described is that it is incapable of latent learning in the absence of reinforcement or a goal. If the goal is just taken away, but the -1 reinforcements are still applied at each step, then the values assigned to each location will tend to $-\infty$. If both are removed, then although the agent will wander about its environment with random gay abandon, it will not pick up anything that could be used to speed subsequent learning. Latent learning experiments with rats in dry mazes prove fairly conclusively that rats running mazes in the absence of rewards and punishments learn almost as much as rats that are reinforced.

One way to solve this problem is suggested by Sutton's DYNA architecture [7]. Briefly, this constructs a map of place × action → next place, and takes steps in the fictitious world constructed from its map in-between taking steps in the real world, as a way of ironing out the computational 'bumps' (*ie* inconsistencies) in the value and action functions.

Instead, it is possible to avoid constructing a complete map by altering the representation of the environment used for learning the prediction function and optimal actions. The section on representations concluded that coarse-coded representations are generally better than punctate ones, since information can be shared between neighbouring points. However, not all neighbouring points are amenable to this sharing, because of discontinuities in the value and action functions. If there were a way of generating a coarse coded representation (generally from a punctate one) that is sensitive to the structure of the task, rather than arbitrarily assigned by the environment, it should provide the base for faster learning still. In this case, neighbouring points should only be coded together if they are not separated by the barrier. The initial exploration would allow the agent to learn this much about the structure of the environment.

Consider a set of units whose job is to predict the future discounted sum of firings of the raw input lines. Using $\mathcal{R}_{BSW}$ during the initial stage of learning when the actions are still random, if the agent is at location (3,3) of the grid, say, then the discounted prediction of how often it will be in (3,4) (*ie* the frequency with which the single unit representing (3,4) will fire) will be high, since this location is close. However, the prediction for (7,11) will be low, because it is very unlikely to get there quickly. Consider the effect of the barrier: locations on opposite sides of it, *eg* (1,6) and (2,6), though close in the Euclidean (or Manhattan) metric on the grid, are far apart in the task. This means that the discounted prediction of how often the agent will be at (1,6) given that it starts at (2,6), will be proportionately lower.

Overall, the prediction units should act like a coarse code, sensitive to the structure of the task. As required, this information about the environment is entirely independent of whether or not the agent is reinforced during its exploration. In fact, the resulting 'map' will be more accurate if it is not, as its exploration will be more random. The output of the prediction units is taken as an additional source of information for the value and action functions.

Since their main aim is to create intelligently distributed representations from punctate ones, it is only appropriate to use these prediction units for $\mathcal{R}_{BSW}$ and $\mathcal{R}_{4x}$. Figure 4 compares average learning curves for $\mathcal{R}_{BSW}$ with and without these ex-

tra mapping units, and with and without 6000 steps of latent learning (LL) in the absence of any reinforcement. A significant improvement is apparent.

Figure 5 shows one set of predictions based on the $\mathcal{R}_{BSW}$ representation[1] after a few un-reinforced iterations. The predictions are clearly fairly well developed and smooth - a predictable exponentially decaying hump. The only deviations from this are at the barrier and along the edges, where the effects of impermeability and immobility are apparent.

Figure 6 shows the same set of predictions but after 2000 reinforced iterations, by which time the agent reaches the goal almost optimally. The predictions degenerate from being roughly radially symmetric (bar the barrier) to being highly asymmetric. Once the agent has learnt how to get to the goal from some location, the path it will follow, and so the locations it will visit from there, is largely fixed. The asymptotic values of the predictions will therefore be 0 for units not on the path, and $\gamma^r$ for those on the path, where $r$ is the number of steps since the agent's start point and $\gamma$ is the discounting factor weighting immediate *versus* distant reinforcement. This is a severe limitation since it implies that the topological information present in the early stages of learning disappears evaporates, and with it almost all the benefits of the prediction units.

## 4   DISCUSSION

Navigation comprises two problems; *where* the agent and the goals in its environment are, and *how* it can get to them. Having some form of cognitive map, as is suggested by the existence of place cells, addresses the first, but leaves open the second. For the case of one goal, the simple TD method described here is one solution.

TD planning methods are clearly robust to changes in the way the input stimulus is represented. Distributed codes, particularly ones that allow for the barrier, make learning faster. This is even true for $\mathcal{R}_A$, which is sensitive to the orientation of the agent. All these results require each location to have a unique representation – Mozer and Bachrach [4] and Chrisley [3] and references therein look at how ambiguities can be resolved using information on the sequence of states the agent traverses.

Since these TD planning methods are totally general, just like dynamical programming, they are unlikely to scale well. Some evidence for this comes from the relatively poor performance of $\mathcal{R}_{4X}$, with its quadrupled input dimension. This puts the onus back either onto dividing the task into manageable chunks, or onto more sophisticated representation.

### Acknowledgements

I am very grateful to Jay Buckingham, Kate Jeffrey, Richard Morris, Toby Tyrell, David Willshaw, and the attendees of the PDP Workshop at Edinburgh, the Connectionist Group at Amherst, and a spatial learning workshop at King's College Cambridge for their helpful comments. This work was funded by SERC.

## Footnotes

[1]Note that these are normalised to a maximum value of 10, for graphical convenience.

# References

[1] Albus, JS (1975). A new approach to manipulator control: the Cerebellar Model Articulation Controller (CMAC). *Transactions of the ASME: Journal of Dynamical Systems, Measurement and Control*, **97**, pp 220-227.

[2] Barto, AG, Sutton, RS & Watkins, CJCH (1989). *Learning and Sequential Decision Making*. Technical Report 89-95, Computer and Information Science, University of Massachusetts, Amherst, MA.

[3] Chrisley, RL (1990). Cognitive map construction and use: A parallel distributed approach. In DS Touretzky, J Elman, TJ Sejnowski, & GE Hinton, editors, *Proceedings of the 1990 Connectionist Models Summer School*. San Mateo, CA: Morgan Kaufmann.

[4] Mozer, MC, & Bachrach, J (1990). Discovering the structure of a reactive environment by exploration. In D Touretzky, editor, *Advances in Neural Information Processing Systems*, *2*, pp 439-446. San Mateo, CA: Morgan Kaufmann.

[5] O'Keefe, J & Nadel, L (1978). *The Hippocampus as a Cognitive Map*. Oxford, England: Oxford University Press.

[6] Sutton, RS (1988). Learning to predict by the methods of temporal difference. *Machine Learning*, **3**, pp 9-44.

[7] Sutton, RS (1990). Integrated architectures for learning, planning, and reacting based on approximating dynamic programming. In *Proceedings of the Seventh International Conference on Machine Learning*. San Mateo, CA: Morgan Kaufmann.

[8] Sutton, RS, & Barto, AG. To appear. Time-derivative models of Pavlovian conditioning. In M Gabriel & JW Moore, editors, *Learning and Computational Neuroscience*. Cambridge, MA: MIT Press.

[9] Watkins, CJCH (1989). *Learning from Delayed Rewards*. PhD Thesis. University of Cambridge, England.

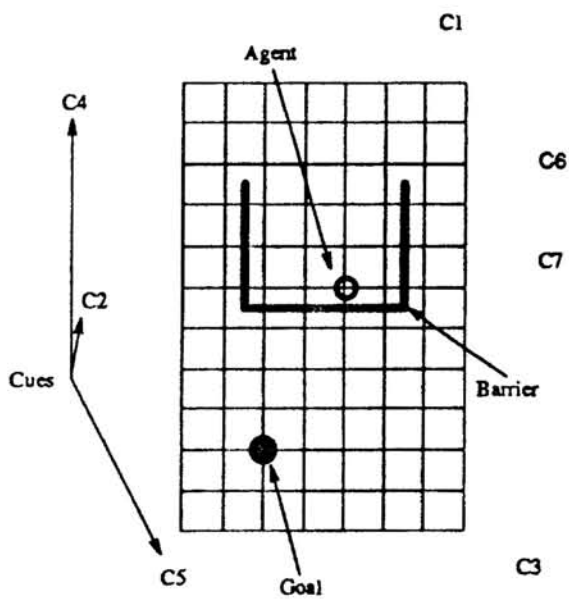

Fig 1: The grid task

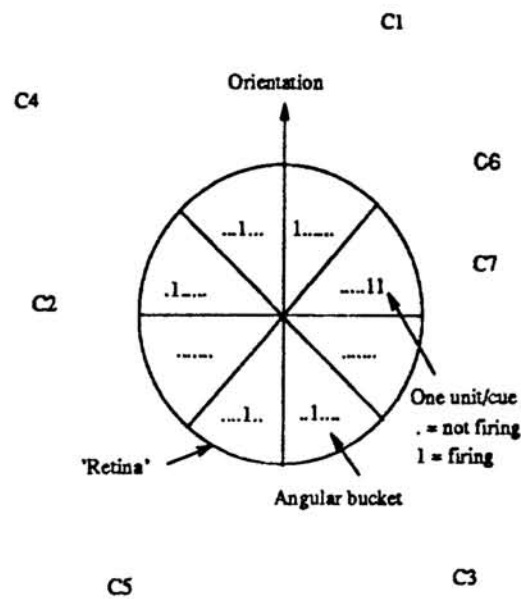

Fig 2: The 'retina' for $\mathcal{R}_A$

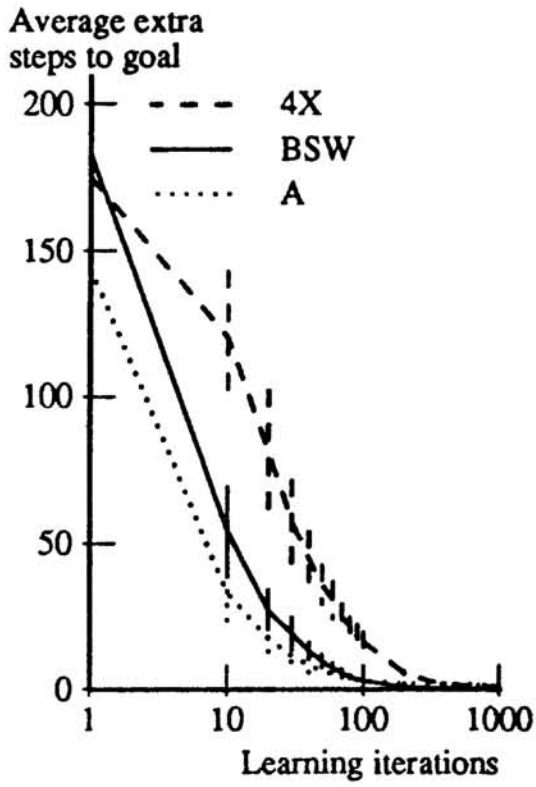

Fig 3: Different representations

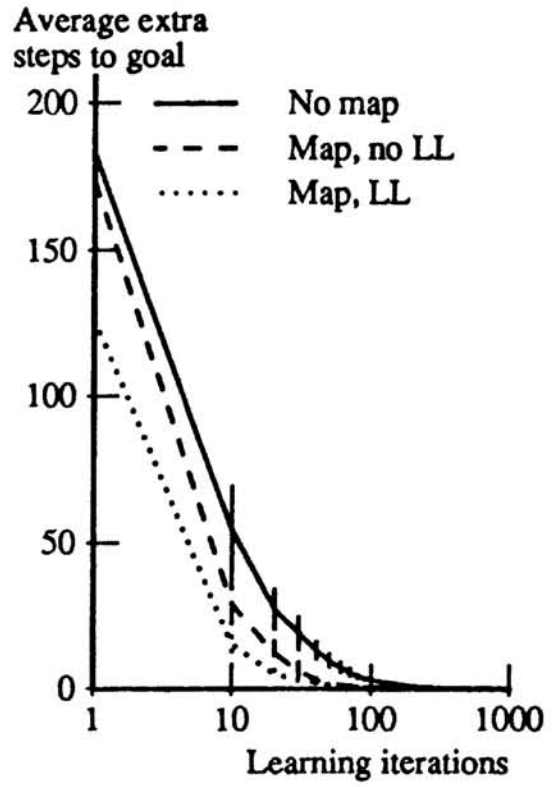

Fig 4: Mapping with $\mathcal{R}_{\text{BSW}}$

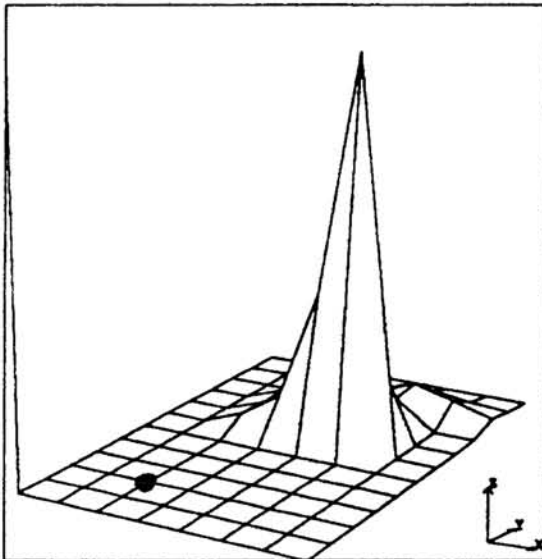

Fig 5: Initial predictions from (5,6)

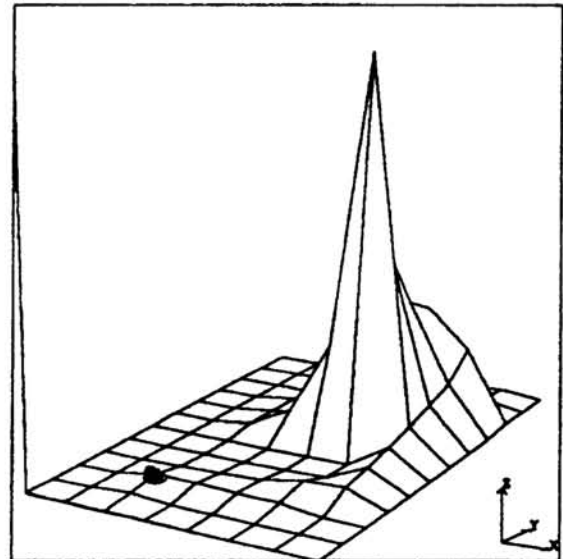

Fig 6: Predictions after 2000 iterations